# Neurometric function analysis of population codes

**Philipp Berens, Sebastian Gerwinn, Alexander S. Ecker and Matthias Bethge**
Max Planck Institute for Biological Cybernetics
Center for Integrative Neuroscience, University of Tübingen
Computational Vision and Neuroscience Group
Spemannstrasse 41, 72076, Tübingen, Germany
`first.last@tuebingen.mpg.de`

## Abstract

The relative merits of different population coding schemes have mostly been analyzed in the framework of stimulus reconstruction using Fisher Information. Here, we consider the case of stimulus discrimination in a two alternative forced choice paradigm and compute neurometric functions in terms of the minimal discrimination error and the Jensen-Shannon information to study neural population codes. We first explore the relationship between minimum discrimination error, Jensen-Shannon Information and Fisher Information and show that the discrimination framework is more informative about the coding accuracy than Fisher Information as it defines an error for any pair of possible stimuli. In particular, it includes Fisher Information as a special case. Second, we use the framework to study population codes of angular variables. Specifically, we assess the impact of different noise correlations structures on coding accuracy in long versus short decoding time windows. That is, for long time window we use the common Gaussian noise approximation. To address the case of short time windows we analyze the Ising model with identical noise correlation structure. In this way, we provide a new rigorous framework for assessing the functional consequences of noise correlation structures for the representational accuracy of neural population codes that is in particular applicable to short-time population coding.

## 1 Introduction

The relative merits of different population coding schemes have mostly been studied (e.g. [1, 12], for a review see [2]) in the framework of stimulus reconstruction (figure 1a), where the performance of a code is judged on the basis of the mean squared error $E[(\theta - \hat{\theta})^2]$. That is, if a stimulus $\theta$ is encoded by a population of $N$ neurons with tuning curves $f_i$, we ask how well, on average, can an estimator reconstruct the true value of the presented stimulus based on the neural responses $\mathbf{r}$, which were generated by the density $p(\mathbf{r}|\theta)$. The average reconstruction error can be written as

$$E_{\theta,\mathbf{r}}[(\theta - \hat{\theta}(\mathbf{r}))^2] = E_\theta[\text{Var}_{\hat{\theta}|\theta}] + E_\theta[b_\theta^2].$$

Here $\text{Var}_{\hat{\theta}|\theta} = E_\mathbf{r}[(\theta - \hat{\theta}(\mathbf{r}))^2|\theta]$ denotes the error variance and $b_\theta = E_\mathbf{r}[\hat{\theta}(\mathbf{r})|\theta] - \theta$ the bias of the estimator $\hat{\theta}$. For the sake of analytical tractability, most studies have employed Fisher Information (FI) (e.g. [1, 12])

$$J_\theta = \left\langle -\frac{\partial^2}{\partial \theta^2} \log p(\mathbf{r}|\theta) \middle| \theta \right\rangle$$

to bound the conditional error variance $\text{Var}_{\hat{\theta}|\theta}$ of an unbiased estimator from below according to the Cramer-Rao bound:

$$\text{Var}_{\hat{\theta}|\theta} \geq \frac{1}{J_\theta}.$$

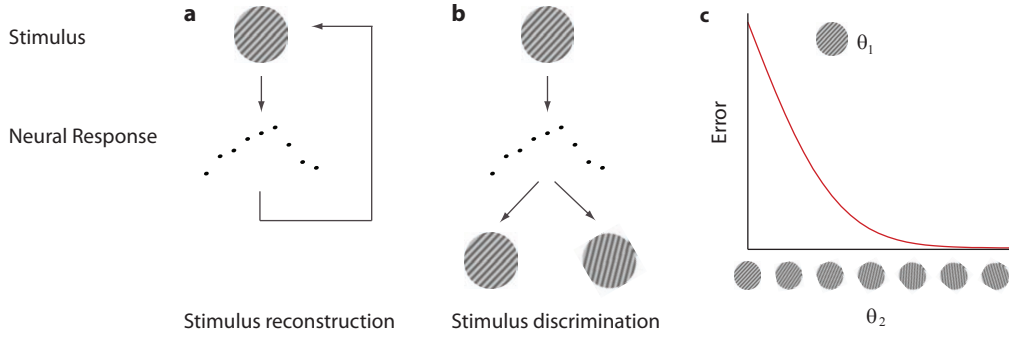

**Figure 1:** Illustration of the two frameworks for studying population codes. **a.** In stimulus reconstruction, an estimator tries to reconstruct the orientation of a stimulus based on a noisy neural response. The quality of a code is based on the average error of this estimator. **b.** In stimulus discrimination, an ideal observer needs to choose one of two possible stimuli based on a noisy neural response (2AFC task). **c.** A neurometric function shows the error $\mathcal{E}$ as a function of $\Delta\theta$, the difference between a reference direction $\theta_1$ and a second direction $\theta_2$. This framework is often used in psychophysical studies.

For the comparison of different coding schemes, it is important that an estimator exists which can actually attain this lower bound. For short time windows and certain types of tuning functions, this may not always be the case [4]. In particular, it is unclear how different population coding schemes affect the fidelity with which a population of binary neurons can encode a stimulus variable.

## 1.1 A new approach for the analysis of population coding

Here we view the population coding problem from a different perspective: We consider the case of stimulus discrimination in a two alternative forced choice paradigm (2AFC, figure 1b) with equally probable stimuli and compute two natural measures of coding accuracy: (1) the minimal discrimination error $\mathcal{E}(\theta_1, \theta_2)$ of an ideal observer classifying a stimulus $s$ based on the response distribution as either being $\theta_1$ or $\theta_2$ and (2) the Jensen-Shannon information $\mathcal{I}_{\mathrm{JS}}$ between the response distributions $p(r|\theta_1)$ and $p(r|\theta_2)$. The minimal discrimination error is achieved by the Bayes optimal classifier $\hat{\theta} = \mathrm{argmax}_s\, p(s|\mathbf{r})$ where $s \in \{\theta_1, \theta_2\}$ and the prior distribution $p(s) = \frac{1}{2}$. It is given by

$$
\begin{aligned}
\mathcal{E}(\theta_1, \theta_2) &= \int \min\left(p(s = \theta_1, \mathbf{r}), p(s = \theta_2, \mathbf{r})\right) \mathrm{d}\mathbf{r} \\
&= \frac{1}{2} \int \min\left(p(\mathbf{r}|\theta_1), p(\mathbf{r}|\theta_2)\right) \mathrm{d}\mathbf{r}
\end{aligned}
\tag{1}
$$

and the Jensen-Shannon Information [13] is defined as

$$
\mathcal{I}_{\mathrm{JS}}(\theta_1, \theta_2) = \frac{1}{2} D_{\mathrm{KL}}\left[p(\mathbf{r}|\theta_1) \| p(\mathbf{r})\right] + \frac{1}{2} D_{\mathrm{KL}}\left[p(\mathbf{r}|\theta_2) \| p(\mathbf{r})\right],
\tag{2}
$$

where $p(\mathbf{r}) = \sum_{s \in \theta_1, \theta_2} p(s) p(\mathbf{r}|s) = \frac{1}{2}(p(\mathbf{r}|\theta_1) + p(\mathbf{r}|\theta_2))$ is the arithmetic average between the two densities, which in our case is the same as the marginal distribution. $D_{\mathrm{KL}}[q_1 \| q_2] = \int q_1(x) \log \frac{q_1(x)}{q_2(x)} \mathrm{d}x$ is the Kullback-Leibler divergence. $\mathcal{I}_{\mathrm{JS}}$ is an interesting measure of coding accuracy since it directly measures the mutual information between the neural responses and the 'class label', i.e. the stimulus identity. By observing a population response pattern $\mathbf{r}$, the uncertainty (in terms of entropy) about the stimulus is reduced by

$$
\mathrm{MI}(\mathbf{r}, s) = \sum_s p(s) \int p(\mathbf{r}|s) \log \frac{p(\mathbf{r}|s)}{\sum_s p(\mathbf{r}|s) p(s)} \mathrm{d}\mathbf{r} = \mathcal{I}_{\mathrm{JS}},
$$

with prior distribution as above. In the following, we will restrict our analysis to the special case of shift-invariant population codes for angular variables and compute neurometric functions $\mathcal{E}(\Delta\theta)$ and $\mathcal{I}_{\mathrm{JS}}(\Delta\theta)$ (figure 1c) by setting $\theta_1 = \theta$ and $\theta_2 = \theta + \Delta\theta$. In the limit of large populations, the dependence of these curves on $\theta$ can be ignored.

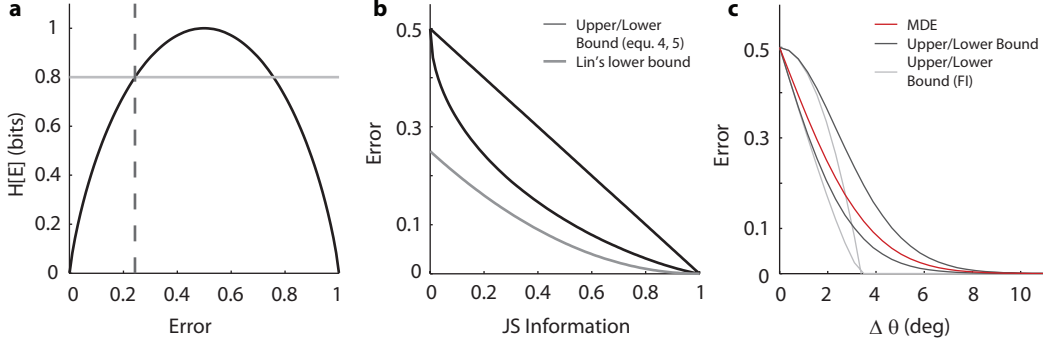

**Figure 2: a.** Illustration of equations 5: The entropy $H[\mathcal{E}]$ (black) intersects $1 - \mathcal{I}_{\text{JS}}$ (grey) at $\mathcal{E}^*$ (dashed). Because of Fano's inequality, $\mathcal{E} > \mathcal{E}^*$. **b.** Functional form of the bounds in equations 4 and 5 (black). Our lower bound is tighter than the lower bound proposed in [13] (grey). **c.** Illustration of the connections between the proposed measures of coding accuracy. Minimal discrimination error $\mathcal{E}(\Delta\theta)$ (red) is shown as a neurometric curve as a function of $\Delta\theta$ and is bounded in terms of the Jensen-Shannon information $\mathcal{I}_{\text{JS}}(\Delta\theta)$ via equations 4 and 5 (black). Fisher Information links to $\mathcal{E}$ via equation 3 and the bounds imposed by $\mathcal{I}_{\text{JS}}$ (grey). This approximation is only valid for small $\Delta\theta$. The computations have been caried out for a population of $N = 50$ neurons, with average correlations $\bar{\rho} = .15$ and correlation structure as in figure 3e.

## 1.2 Computing $\mathcal{E}$ and $\mathcal{I}_{\text{JS}}$

While the integrals in equation (1) and (2) often cannot be solved, they are relatively easy to evaluate numerically using Monte-Carlo techniques [10]. For the minimal discrimination error, we use

$$\mathcal{E}(\Delta\theta) = \frac{1}{2} \int \min\left(p(\mathbf{r}|\theta), p(\mathbf{r}|\theta + \Delta\theta)\right) \mathrm{d}\mathbf{r}$$

$$\approx \frac{1}{2} \sum_{i=1}^{M} \min\left(p(\mathbf{r}^{(i)}|\theta), p(\mathbf{r}^{(i)}|\theta + \Delta\theta)\right) / p(\mathbf{r}^{(i)}),$$

where $\mathbf{r}^{(i)}$ is one of $M$ samples, drawn from the mixture distribution $p(\mathbf{r}) = \frac{1}{2}\left(p(\mathbf{r}|\theta) + p(\mathbf{r}|\theta + \Delta\theta)\right)$. To approximate $\mathcal{I}_{\text{JS}}$, we evaluate each $D_{\text{KL}}$ term separately as

$$D_{\text{KL}}\left[p(\mathbf{r}|\theta)\|p(\mathbf{r})\right] = \int p(\mathbf{r}|\theta) \log \frac{p(\mathbf{r}|\theta)}{p(\mathbf{r})} \mathrm{d}\mathbf{r}$$

$$\approx \frac{1}{M} \sum_{i=1}^{M} \log p(\mathbf{r^{(i)}}|\theta) - \log p(\mathbf{r}^{(i)})$$

where we draw samples $\mathbf{r}^{(i)}$ from $p(\mathbf{r^{(i)}}|\theta)$. We use an analogous expression for $D_{\text{KL}}\left[p(\mathbf{r}|\theta + \Delta\theta)\|p(\mathbf{r})\right]$ and plug these estimates into equation 2. This scheme provides consistent estimates of the desired quantities. For all simulations below we used $M = 10^5$ samples.

## 2 Links between the proposed measures

In this section, we link the Fisher Information $J_\theta$ of a population code $p(\mathbf{r}|\theta)$ to the minimum discrimination error $\mathcal{E}(\Delta\theta)$ and the Jensen-Shannon Information $\mathcal{I}_{\text{JS}}(\Delta\theta)$ in the 2AFC paradigm. First, we link Fisher Information to Jensen-Shannon information $\mathcal{I}_{\text{JS}}$. Second, we bound the minimum discrimination error in terms of the Jensen-Shannon information.

### 2.1 From Fisher Information to Jensen-Shannon Information

In order to obtain a relationship between $\mathcal{I}_{\text{JS}}$ and Fisher Information, we use an expression already derived in [7], where $p(\mathbf{r}|\theta + \Delta\theta)$ is expanded up to second order in $\Delta\theta$, which yields:

$$\mathcal{I}_{\text{JS}}(\Delta\theta) \approx \frac{1}{8}(\Delta\theta)^2 J_\theta. \tag{3}$$

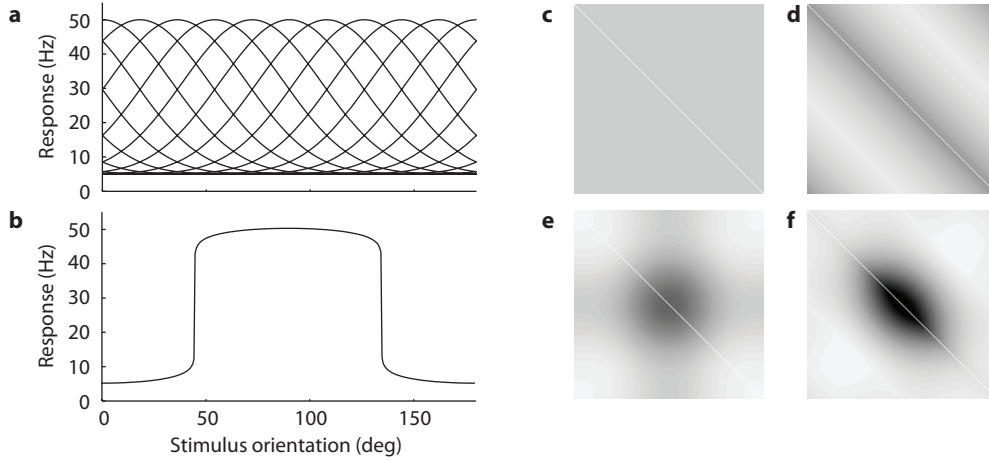

**Figure 3:** Illustration of the model. *Tuning functions:* **a.** Cosine-type tuning functions with rates between 5 and 50 Hz. **b.** Box-like tuning function with matched minimal and maximal firing rates. Cosine tuning function resembles the orientation tuning functions of many cortical neurons. They are characterized by approximately constant Fisher Information independent of the stimulus orientation. Box-like tuning functions, in contrast, have non-constant Fisher Information due to their steep non-linearity. They have been shown to exhibit superior performance over cosine-like tuning functions with respect to the mean squared error [4]. *Correlation matrices:* **c.** stimulus-independent, no limited range (SI, $\alpha = \infty$) , **d.** stimulus-independent, limited range (SI, $\alpha = 2$), **e.** stimulus-dependent, no limited range (SD, $\alpha = \infty$), **f.** stimulus-dependent, limited range (SD, $\alpha = 2$)

Therefore, Fisher Information provides a good approximation of the Jensen-Shannon Information for sufficiently small $\Delta\theta$.

## 2.2   From Jensen-Shannon Information to Minimal Discrimination Error

The minimal discrimination error $\mathcal{E}(\Delta\theta)$ of an ideal observer is bounded from above and below in terms of $\mathcal{I}_{JS}(\Delta\theta)$. An upper bound derived by [13] is given by

$$\mathcal{E}(\Delta\theta) \leq \frac{1}{2} - \frac{1}{2}\mathcal{I}_{JS}(\Delta\theta). \tag{4}$$

Next, we derive a new lower bound on $\mathcal{E}$, which is tighter than a bound derived by Lin [13]. To this end, we observe that from Fano's inequality [8] it follows that

$$
\begin{aligned}
H\left[\mathcal{E}\right] \geq \quad & H[s|\mathbf{r}] - \mathcal{E}\log(|s| - 1) \\
= \quad & H[s|\mathbf{r}] \\
= \quad & H[s] - \mathrm{MI}[\mathbf{r}, s] \\
= \quad & 1 - \mathcal{I}_{JS}(\Delta\theta),
\end{aligned}
\tag{5}
$$

where $H[\mathcal{E}]$ is the entropy of a Bernoulli distribution with $p = \mathcal{E}$. The equality from first to second line follows as the number of stimuli or classes $|s| = 2$. Since the entropy is monotonic in $\mathcal{E}$ on the interval $[0, 0.5]$, we have the lower bound $\mathcal{E} \geq \mathcal{E}^*$, where $\mathcal{E}^*$ is chosen such that equality holds. For an illustration, see figure 2a. The shape of both bounds, as well as Lin's lower bound, are illustrated in figure 2b.

In figure 2c we show the minimal discrimination error for a population code (red) together with the upper and lower bound (black) obtained by inserting $\mathcal{I}_{JS}(\Delta\theta)$ into equations 4 and 5. Both bounds follow nicely the neurometric function $\mathcal{E}(\Delta\theta)$. For comparison, we also show the upper and lower bound obtained by plugging Fisher Information into equation 3 and computing the bounds 4 and 5 based on this approximation of $\mathcal{I}_{JS}(\Delta\theta)$ (grey). Clearly, the approximation is valid for small $\Delta\theta$ and becomes successively worse for large ones.

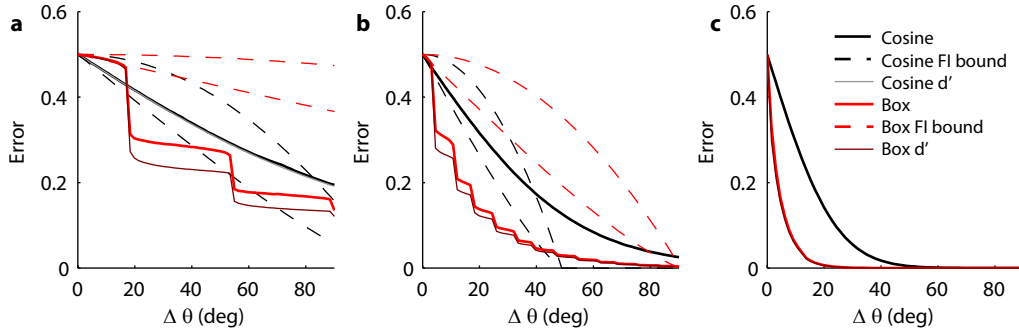

**Figure 4:** Comparison of box-like (red) vs. cosine (black) tuning functions in short-term population codes of **a.** $N = 10$ **b.** $N = 50$ **c.** $N = 250$ independent neurons. Although box-like tuning functions are much broader than cosine tuning functions, $\mathcal{E}_{box}$ lies usually below $\mathcal{E}_{cos}$. For the cosine case, FI (dashed, approximation as in figure 2c and $\mathcal{E}_{d'}$ (grey) provide accurate accounts of coding accuracy. In contrast, FI grossly overestimates the discrimination error for box-like tuning functions in small and medium sized populations. In this case, $\mathcal{E}_{d'}$ is only a good approximation of $\mathcal{E}$ in the range where $\Delta\theta$ is small (dark red). Beyond this point, it underestimates $\mathcal{E}$ (a,b). For $N = 250$, bounds are not shown for clarity but they capture the true beaviour of $\mathcal{E}$ better than in figure 4a and b.

## 2.3 Previous work

Only a small number of studies on neural population coding have used other measures than Fisher Information [18, 3, 6, 4]. Two approaches are most closely related to ours: Snippe and Koenderink [18] and Averbeck and Lee [3] used a measure analogous to the sensitivity index $d'$

$$(d')^2 = \Delta\mu\Sigma^{-1}\Delta\mu \qquad (6)$$
$$\Delta\mu := \mathbf{f}(\theta + \Delta\theta) - \mathbf{f}(\theta)$$

as a measure of coding accuracy. While Snippe and Koenderink have considered only the limit $\Delta\theta \to 0$, Averbeck and Lee evaluated equation 6 for finite $\Delta\theta$ using $\Sigma = \frac{1}{2}(\Sigma_\theta + \Sigma_{\theta+\Delta\theta})$ and converted $d'$ to a discrimination error $\mathcal{E}_{d'} = 1 - \text{erf}(d'/2)$. This approximation is exact only if the class conditional distribution $p(\mathbf{r}|\theta)$ is Gaussian with fixed covariance $\Sigma_\theta = \Sigma$ for all $\Delta\theta$. In that particular case, the entire neurometric function is fully determined by the Fisher Information [9]:

$$d' = (\Delta\theta)\sqrt{J_\theta} = (\Delta\theta)\,J_{\text{mean}}$$

$J_{\text{mean}}$ is the linear part of the Fisher Information (cf. equation 7). In the general case, it is not obvious what aspects of the quality of a population code are captured by the above measure. Therefore, both Fisher Information and the class-conditional second-order approximation used by Averbeck and Lee have shortcomings: The latter does not account for information originating from changes in the covariance matrix as is quantified by $J_{\text{cov}}$ (cf. equation 7). Fisher Information, on the other hand, can be quite uninformative about the coding accuracy of the population, especially when the tuning functions are highly nonlinear (see figure 3) or noise is large, as in these cases it is not certain whether the Cramer-Rao bound can actually be attained [4]. The examples studied in the next section demonstrate how these shortcomings can be overcome using the minimal discrimination error (equation 1).

## 3 Results

After describing the population model used in this study, we will illustrate in a simple example, how our proposed framework is more informative than previous approaches. Second, we will investigate how different noise correlations structures impact population coding on different timescales.

## 3.1 The population model

In this section, we describe in detail the population model used in the remainder of the study. To facilitate comparability, we closely follow the model used in a recent study by Josic et al. [12]

where applicable. We consider a population of $N$ neurons tuned to orientation, where the firing rate of neuron $i$ follows an average tuning profile $f_i(\theta)$ with (a) a cosine-like shape

$$f_i(\theta) = \lambda_1 + \lambda_2 a^k(\theta - \phi_i)$$

with $k = 1$ in section 3.2 and $k = 6$ in section 3.3 and $a(\phi) = \frac{1}{2}(1 + \cos(\phi))$ or (b) a box-like shape

$$f_i(\theta) = \left(|\cos(\theta - \phi_i)|^{\frac{1}{j}} \cdot \operatorname{sgn}\cos(\theta - \phi_i) + 1\right) \cdot \frac{\lambda_2}{2} + \lambda_1.$$

Here, $\phi_i$ is the preferred orientation of neuron $i$ and we use $j = 12$. We consider two scenarios:

1. Long-term coding: $r(\theta) \sim \mathcal{N}\left(f(\theta), \Sigma(\theta)\right)$, where the trial-to-trial fluctuations are assumed to be normally distributed with mean $f(\theta)$ and covariance matrix $\Sigma(\theta)$.

2. Short-term coding: $r(\theta) \sim I\left(f(\theta), \Sigma(\theta)\right)$, where $r_i \in \{0, 1\}$ and $I(\mu, \Sigma)$ is the maximum entropy distribution consistent with the constraints provided by $\mu$ and $\Sigma$, the Ising model [16]. That is, for short-term population coding, we assume the population acitivity to be binary with each neuron either emitting one spike or none. The parameters of the Ising model were computed using gradient descent on the log likelihood.

Following Josic et al. [12], we model the stimulus-dependent covariance matrix as $\Sigma_{ij}(\theta) = \delta_{ij}v_i(\theta) + (1 - \delta_{ij})\rho_{ij}(\theta)\sqrt{v_i(\theta)v_j(\theta)}$, where $v_i(\theta)$ is the variance of cell $i$ and $\rho_{ij}(\theta)$ the correlation coefficient. For long-term coding, we set $v_i(\theta) = f_i(\theta)$ and for short-term coding, we set $v_i(\theta) = f_i(\theta)(1 - f_i(\theta))$. We allow for both stimulus and spatial influences on $\rho$ by setting $\rho_{ij}(\theta) = \sigma_{ij}(\theta)c(\phi_i - \phi_j)$, where $\phi_i$ is the preferred orientation of neuron $i$. The function $s$ models the influence of the stimulus, while the function $c$ models the spatial component of the correlation structure. We use $\sigma_{ij}(\theta) = \sigma_i(\theta)\sigma_j(\theta)$, where $\sigma_i(\theta) = \kappa_1 + \kappa_2 a^2(\theta)$. We set $c(\phi_i - \phi_j) = C\exp\left(-|\phi_i - \phi_j|/\alpha\right)$, where $\alpha$ controls the length of the spatial decay. To obtain a desired mean level of correlation $\bar{\rho}$, we use the method described in [12].

## 3.2 Minimum discrimination error is more informative than Fisher Information

As has been pointed out in [4], the shape of unimodal tuning functions can strongly influence the coding accuracy of population codes of angular variables. In particular, box-like tuning functions can be superior to cosine tuning functions. However, numerical evaluation of the minimum mean squared error for angular variables is much more difficult than the evaluation of the minimal discrimination error proposed here, and the above claim has only been verified up to $N = 20$ neurons.

Here we compute the full neurometric functions for $N = 10, 50, 250$ binary neurons (figure 4). In this way, we show that the advantage of box-like tuning functions also holds for large numbers of neurons (compare red and black curves in figure 4 a-c). In addition, we note that Fisher Information does not provide an accurate account of the performance of box-like tuning functions: it fails as soon as the nonlinearity in the tuning functions becomes effective and overestimates the true minimal discrimination error $\mathcal{E}$. Similarly, the approximate neurometric functions $\mathcal{E}_{d'}(\Delta\theta)$ obtained from equation 6 do not capture the shape of neurometric functions $\mathcal{E}(\Delta\theta)$ but underestimate the minimal discrimination error. In contrast, the deviation between both curves stays rather small for cosine tuning functions.

## 3.3 Stimulus-dependent correlations have opposite effects for long- and short-term population coding

The shape of the noise covariance matrix $\Sigma_\theta$ can strongly influence the coding fidelity of a neural population. In order to evaluate these effects it is important to take differences in the noise covariance for different stimuli into account. In this section, we will use our new framework to study different noise correlation structures for short- and long-term population coding.

Previous studies so far have investigated the effect of noise correlations in the long-term case: Most studies assumed $p(\mathbf{r}|\theta)$ to follow a multivariate Gaussian distribution, so that firing rates $\mathbf{r}|\theta \sim \mathcal{N}\left(\mathbf{f}(\theta), \Sigma(\theta)\right)$ (for detailed description of the population model see section 3.1). In this case, the

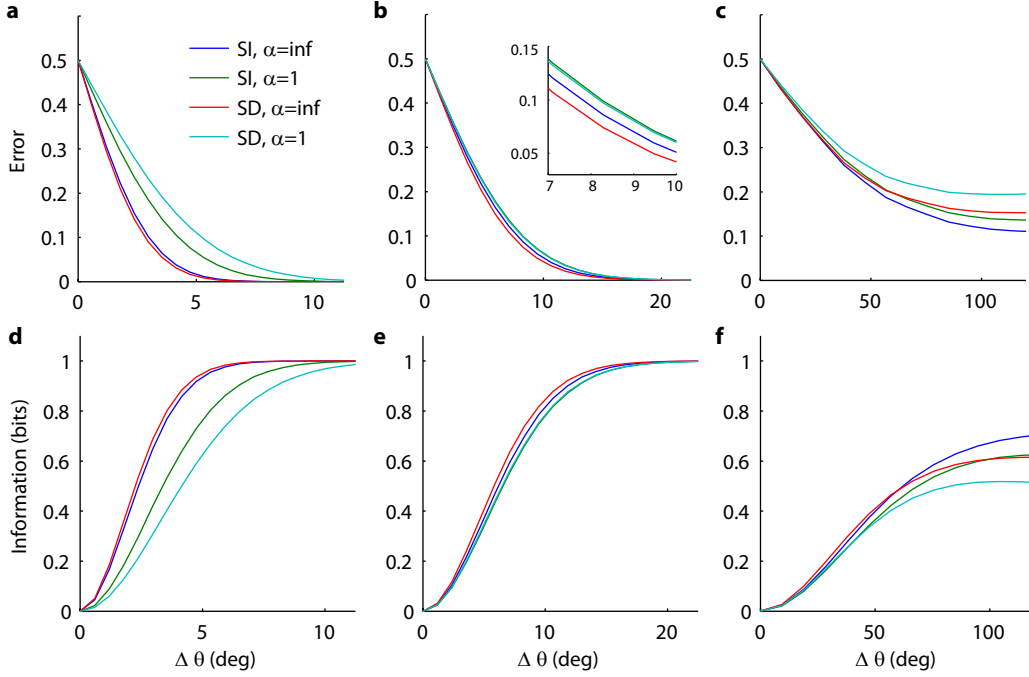

**Figure 5:** Neurometric functions $\mathcal{E}(\Delta\theta)$ (a-c) and $\mathcal{I}_{JS}(\Delta\theta)$ (d-f) for four different noise correlation structures. **a.** and **d.** Large population ($N = 100$) and long-term coding. **b.** and **e.** Medium sized population ($N = 15$) and long-term coding. The inset is a magnification for clarity. **c.** and **f.** Medium sized population ($N = 15$) and short-term coding. The impact of stimulus-dependent noise correlations in the absence of limited range correlations changes from b/e to c/f (red line). While they are beneficial in long-term coding, they are beneficial in short-term coding only for close angles. The exact point of this transition is not the same for $\mathcal{E}$ and $\mathcal{I}_{JS}$, since they are only related via the bounds described in section 2.2. Note that the scale of the x-axis varies.

FI of the population takes a particularly simple form. It can be decomposed into:

$$J_\theta = J_{\mathrm{mean}} + J_{\mathrm{cov}}$$

$$J_{\mathrm{mean}} = \mathbf{f}'^\top \Sigma^{-1} \mathbf{f}', \qquad J_{\mathrm{cov}} = \frac{1}{2}\mathrm{Tr}[\Sigma'\Sigma^{-1}\Sigma'\Sigma^{-1}], \tag{7}$$

where we omit the dependence on $\theta$ for clarity and $\mathbf{f}'$, $\Sigma'$ are the derivatives of $\mathbf{f}$ and $\Sigma$ with respect to $\theta$. $J_{\mathrm{mean}}$, $J_{\mathrm{cov}}$ are the Fisher information, when either only the mean or only the covariance are assumed to depend on $\theta$. For this case, various studies have investigated noise structures where correlations were either uniform across the population (figure 3c) or their magnitude decayed with difference in preferred orientations (figure 3d), 'limited range structure' or 'spatial decay', see e.g. [1]). Only recently have stimulus-dependent correlations been analyzed in terms of Fisher information [12]. Josic et al. find that in the absence of limited range correlations, stimulus-dependent noise correlations (figure 3e) are beneficial for a population code, while in their presence (figure 3f), they are detrimental.

We first compute the neurometric functions $\mathcal{E}(\Delta\theta)$ and $\mathcal{I}_{JS}(\Delta\theta)$ for a population of 100 neurons in the case of long-term coding with a Gaussian noise model for the four possible noise correlation structures (figure 5a). We corroborate the results of Josic et al. in that we find that the lowest $\mathcal{E}$ or the highest $\mathcal{I}_{JS}$ is achieved for a population with stimulus-dependent noise correlations and no limited range structure, while a population with stimulus-dependent noise correlations in the presence of spatial decay performs worst. Spatially uniform correlations (figure 3c) provide almost as good a code as the best coding scheme.

Next, we directly compare long- and short-term population coding in a population of 15 neurons[1]. For short-term coding, we assume that the population activity is of binary nature, i.e. each neuron spikes at most once. Again, we compute neurometric functions $\mathcal{E}(\Delta\theta)$ and $\mathcal{I}_{\text{JS}}(\Delta\theta)$ for all four possible correlation structures. The results for long-term coding do not differ between large and small populations (figure 5b), although relative differences between different coding schemes are less prominent. In contrast, we find that the beneficial impact of stimulus-dependent correlations in the absence of limited range structure reverses in short-term codes for large $\Delta\theta$ (figure 5c).

## 4 Discussion

In this paper, we introduce the computation of neurometric functions as a new framework for studying the representational accuracy of neural population codes. Importantly, it allows for a rigorous treatment of nonlinear population codes (e.g. box-like tuning functions) and noise correlations for non-Gaussian noise models. This is particularly important for binary population codes on timescales where neurons fire at most one spike. Such codes are of special interest since psychophysical experiments have demonstrated that efficient computations can be performed in cortex on short time scales [19]. Previous studies have mostly focussed on long-term population codes, since in this case it is possible to study many question analytically using Fisher Information. Although the structure of neural population acitivity on short timescales has recently attracted much interest [16, 17, 15], population codes for binary population activity and, in particular, the impact of different noise correlation structures on such codes are not well understood. In contrast to previous work [14], neurometric function analysis allows for a comprehensive treatment of both short- and long-term population codes in a single framework. In section 3.3, we have started to study population codes on short timescales and found important differences in the effect of noise correlations between short- and long-term population codes. In the future, we will extend these results to much larger populations adapting new techniques for approximate fitting of Ising models [15].

The example discussed in section 3.2 demonstrates that neurometric functions can provide additional information compared to Fisher Information: While Fisher Information is a single number for each potential population code, neurometric functions in terms of $\mathcal{E}$ or $\mathcal{I}_{\text{JS}}$ assess the coding quality for each pair of stimuli. This also enables us to detect effects like the dependence of the relative performance of different population codes on $\Delta\theta$ as shown in figure 5 c and f. We can furthermore easily extend the framework to take unequal prior probabilities into account. In equations 1 and 2 we have assumed equal prior probabilities $p(\theta_1) = p(\theta_2) = \frac{1}{2}$. Both $\mathcal{E}$ and $\mathcal{I}_{\text{JS}}$, however, are also well defined if this is not the case.

The framework of stimulus discrimination in a 2AFC task has long been used in psychophysical and neurophysiological studies for measuring the accuracy of orientation coding in the visual system (e.g. [5, 21]). It is therefore appealing to use the same framework also in theoretical investigations on neural population coding since this facilitates the comparison with experimental data. Furthermore, it allows studying population codes for categorial variables since, in contrast to Fisher Information, it does not require the variable of interest to be continuous. This is of advantage, as many neurophysiological studies investigate the encoding of categories, such as objects [11] or numbers [20].

**Acknowledgments**

We thank A. Tolias and J. Cotton for discussions. This work has been supported by the Bernstein award to MB (BMBF; FKZ: 01GQ0601) and a scholarship of the German National academic foundation to PB.

## Footnotes

[1]We are limited in the number of neurons as fitting the required Ising model is computationally very expensive. For the present purpose, we chose $N = 15$, which is sufficient to demonstrate our point.

# References

[1] L. F. Abbott and Peter Dayan. The effect of correlated variability on the accuracy of a population code. *Neural Comp.*, 11(1):91–101, 1999.

[2] B. B. Averbeck, P. E. Latham, and A. Pouget. Neural correlations, population coding and computation. *Nat Rev Neurosci*, 7(5):358–366, 2006.

[3] B. B. Averbeck and D. Lee. Effects of noise correlations on information encoding and decoding. *J Neurophysiol*, 95(6):3633–3644, 2006.

[4] M. Bethge, D. Rotermund, and K. Pawelzik. Optimal Short-Term population coding: When fisher information fails. *Neural Comp.*, 14(10):2317–2351, 2002.

[5] A. Bradley, B. C. Skottun, I. Ohzawa, G. Sclar, and R. D. Freeman. Visual orientation and spatial frequency discrimination: a comparison of single neurons and behavior. *J Neurophysiol*, 57(3):755–772, 1987.

[6] N. Brunel and J. P. Nadal. Mutual information, fisher information, and population coding. *Neural Computation*, 10(7):1731–1757, 1998.

[7] M. Casas, P. W. Lamberti, A. Plastino, and A. R. Plastino. Jensen-Shannon divergence, fisher information, and wootters' hypothesis. *Arxiv preprint quant-ph/0407147*, 2004.

[8] T. M. Cover and J. A. Thomas. *Elements of Information Theory*. Wiley-Interscience, 2006.

[9] P. Dayan and L. F. Abbott. *Theoretical neuroscience: Computational and mathematical modeling of neural systems*. MIT Press, 2001.

[10] J.R. Hershey and P.A. Olsen. Approximating the kullback leibler divergence between gaussian mixture models. In *Acoustics, Speech and Signal Processing, 2007. ICASSP 2007. IEEE International Conference on*, volume 4, pages IV–317–IV–320, 2007.

[11] C. P. Hung, G. Kreiman, T. Poggio, and J. J. DiCarlo. Fast readout of object identity from macaque inferior temporal cortex. *Science*, 310(5749):863–866, 2005.

[12] K. Josic, E. Shea-Brown, B. Doiron, and J. de la Rocha. Stimulus-dependent correlations and population codes. *Neural Computation*, 21(10):2774–2804, 2009.

[13] J. Lin. Divergence measures based on the shannon entropy. *Information Theory, IEEE Transactions on*, 37(1):145–151, 1991.

[14] S. Panzeri, A. Treves, S. Schultz, and E. T. Rolls. On decoding the responses of a population of neurons from short time windows. *Neural Computation*, 11(7):1553–1577, 1999.

[15] Y. Roudi, J. Tyrcha, and J. Hertz. The ising model for neural data: Model quality and approximate methods for extracting functional connectivity. *Phys. Rev. E*, 79:051915, February 2009.

[16] E. Schneidman, M. J. Berry, R. Segev, and W. Bialek. Weak pairwise correlations imply strongly correlated network states in a neural population. *Nature*, 440(7087):1007–1012, 2006.

[17] J. Shlens, G. D. Field, J. L. Gauthier, M. Greschner, A. Sher, A. M. Litke, and E. J. Chichilnisky. The structure of Large-Scale synchronized firing in primate retina. *Journal of Neuroscience*, 29(15):5022, 2009.

[18] H. Snippe and J. Koenderink. Information in channel-coded systems: correlated receivers. *Biological Cybernetics*, 67(2):183–190, June 1992.

[19] S. Thorpe, D. Fize, and C. Marlot. Speed of processing in the human visual system. *Nature*, 381(6582):520–522, 1996.

[20] O. Tudusciuc and A. Nieder. Neuronal population coding of continuous and discrete quantity in the primate posterior parietal cortex. *Proceedings of the National Academy of Sciences of the United States of America*, 104(36):14513–8, 2007.

[21] P. Vazquez, M. Cano, and C. Acuna. Discrimination of line orientation in humans and monkeys. *J Neurophysiol*, 83(5):2639–2648, 2000.

